# A Theory for Neural Networks with Time Delays

**Bert de Vries**
Department of Electrical Engineering
University of Florida, CSE 447
Gainesville, FL 32611

**Jose C. Principe**
Department of Electrical Engineering
University of Florida, CSE 444
Gainesville, FL 32611

## Abstract

We present a new neural network model for processing of temporal patterns. This model, the *gamma neural model*, is as general as a convolution delay model with arbitrary weight kernels w(t). We show that the gamma model can be formulated as a (partially prewired) additive model. A temporal hebbian learning rule is derived and we establish links to related existing models for temporal processing.

## 1    INTRODUCTION

In this paper, we are concerned with developing neural nets with short term memory for processing of temporal patterns. In the literature, basically two ways have been reported to incorporate short-term memory in the neural system equations. The first approach utilizes reverberating (self-recurrent) units of type $\frac{dx}{dt} = -a\sigma(x) + e$, that hold a trace of the past neural net states x(t) or the input e(t). Elman (1988) and Jordan (1986) have successfully used this approach. The disadvantage of this method is the lack of weighting flexibility in the temporal domain, since the system equations are described by first order dynamics, implementing a recency gradient (exponential for linear units).

The second approach involves explicit inclusion of delays in the neural system equations. A general formulation for this type requires a time-dependent weight matrix W(t). In such a system, multiplicative interactions are substituted by temporal convolution operations, leading to the following system equations for an additive *convolution model* -

$$\frac{dx}{dt} = \int_0^t W(t-s)\,\sigma(x(s))\,ds + e. \tag{1}$$

Due to the complexity of general convolution models, only strong simplifications of the weight kernel have been proposed. Lang et. al. (1990) use a delta function kernel, $W(t) = \sum_{k=0}^{K} W_k \delta(t-t_k)$, which is the core for the Time-Delay-Neural-Network (TDNN). Tank and Hopfield (1987) prewire W(t) as a weighted sum of dispersive delay kernels, $W(t) = \sum_{k=0}^{K} W_k (\frac{t}{\tau_k})^k e^{k(1-\frac{t}{\tau_k})} = \sum_{k=0}^{K} W_k h_k(t, \tau_k)$. The kernels $h_k(t, \tau_k)$ are the integrands of the gamma function. Tank and Hopfield described a one-layer system for classification of isolated words. We will refer to their model as a Concentration-In-Time-Network (CITN). The system parameters were non-adaptive, although a Hebbian rule equivalent in functional differential equation form was suggested.

In this paper, we will develop a theory for neural convolution models that are expressed through a sum of gamma kernels. We will show that such a gamma neural network can be reformulated as a (Grossberg) additive model. As a consequence, the substantial learning and stability theory for additive models is directly applicable to gamma models.

## 2    THE GAMMA NEURAL MODEL - FORMAL DERIVATION

Consider the N-dimensional convolution model -

$$\frac{dx}{dt} = -ax + W_0 y + \int_0^t ds\, W(t-s)\, y(s) + e, \qquad (2)$$

where x(t), y(t)=σ(x) and e(t) are N-dimensional signals; $W_0$ is NxN and W(t) is NxNx $[0, \infty]$. The weight matrix $W_0$ communicates the direct neural interactions, whereas W(t) holds the weights for the delayed neural interactions. We will now assume that W(t) can be written as a linear combination of normalized gamma kernels, that is, -

$$W(t) = \sum_{k=1}^{K} W_k g_k(t), \qquad (3)$$

where -

$$g_k(t) = \frac{\mu^k}{(k-1)!} t^{k-1} e^{-\mu t}, \qquad (4)$$

where μ is a decay parameter and k a (lag) order parameter. If W(t) decays exponentially to zero for $t \to \infty$, then it follows from the completeness of Laguerre polynomials that this approximation can be made arbitrarily close (Cohen et. al.,

1979). In other words, for all physical plausible weight kernels there is a K such that W(t) can be expressed as ( 3), ( 4). The following properties hold for the gamma kernels $g_k(t)$ -

- [1] The gamma kernels are related by a set of linear homogeneous ODEs -

$$\frac{dg_1}{dt} = -\mu g_1$$

$$\frac{dg_k}{dt} = -\mu g_k + \mu g_{k-1}, \quad k=2,..,K$$

(5)

- [2] The peak value $(\frac{dg_k}{dt} = 0)$ occurs at $t_p = \frac{k-1}{\mu}$.

- [3] The area of the gamma kernels is a normalized, that is, $\int_0^\infty ds g_k(s) = 1$.

Substitution of ( 3) into ( 2) yields -

$$\frac{dx}{dt} = -ax + \sum_{k=0}^{K} W_k y_k + e,$$

(6)

where we defined $y_0(t) = y(t)$ and the gamma state variables -

$$y_k(t) = \int_0^t ds g_k(t-s) y_0(s), \quad k=1,..,K.$$

(7)

The gamma state variables hold memory traces of the neural states $y_0(t)$. The important question is how to compute $y_k(t)$. Differentiating ( 7) using Leibniz' rule yields -

$$\frac{dy_k}{dt} = \int_0^t \frac{\partial}{\partial t} g_k(t-s) y(s) ds + g_k(0) y(t).$$

(8)

We now utilize gamma kernel property [1] (eq. ( 5)) to obtain -

$$\frac{dy_k}{dt} = \int_0^t [-\mu g_k(t-s) + \mu g_{k-1}(t-s)] y(s) ds + g_k(0) y(t)$$

(9)

Note that since $g_k(0) = 0$ for $k \geq 2$ and $g_1(0) = \mu$, ( 9) evaluates to -

$$\frac{dy_k}{dt} = -\mu y_k + \mu y_{k-1}, \quad k=1,..,K.\tag{10}$$

The gamma model is described by ( 6) and ( 10). This extended set of ordinary differential equations (ODEs) is equivalent to the convolution model, described by the set of functional differential equations ( 2), ( 3) and ( 4).

It is a valid question to ask whether the system of ODEs that describes the gamma model can still be expressed as a neural network model. The answer is affirmative, since the gamma model can be formulated as a regular (Grossberg) additive model.

To see this, define the N(K+1)-dimensional augmented state vector $X = \begin{bmatrix} x \\ y_1 \\ | \\ y_K \end{bmatrix}$, the

neural output signal $Y = \begin{bmatrix} \sigma(x) \\ y_1 \\ | \\ y_K \end{bmatrix}$, an external input $E = \begin{bmatrix} e \\ 0 \\ | \\ 0 \end{bmatrix}$, a diagonal matrix

of decay parameters $M = \begin{bmatrix} a & & 0 \\ & \mu & \\ & & \ddots \\ 0 & & \mu \end{bmatrix}$ and the weight (super)matrix

$\Omega = \begin{bmatrix} W_0 & W_1 & \cdots & W_K \\ \mu & & & \\ & \ddots & & 0 \\ 0 & & \mu & 0 \end{bmatrix}$. Then the gamma model can be rewritten in the following

form -

$$\frac{dX}{dt} = -MX + \Omega Y + E,\tag{11}$$

the familiar Grossberg additive model.

## 3     HEBBIAN LEARNING IN THE GAMMA MODEL

The additive model formulation of the gamma model allows a direct generalization

of learning techniques to the gamma model. Note however that the augmented state vector X contains the gamma state variables $y_1,...,y_K$, basically (dispersively) delayed neural states. As a result, although associative learning rules for conventional additive models only encode the simultaneous correlation of neural states, the gamma learning rules are able to encode temporal associations as well. Here we present Hebbian learning for the gamma model.

The Hebbian postulate is often mathematically translated to a learning rule of the form $\frac{dW}{dt} = \eta x(t) y^T(t)$, where $\eta$ is a learning rate constant, $x$ the neural activation vector and $y^T$ the neuron output signal vector. This procedure is not likely to encode temporal order, since information about past states is not incorporated in the learning equations.

Tank and Hopfield (1987) proposed a generalized Hebbian learning rule with delays that can be written as -

$$\frac{dW}{dt} = \eta x(t) \left[ \int_0^t ds\, g(s) y(t-s) \right]^T , \qquad (12)$$

where $g(s)$ is a normalized delay kernel. Notice that (12) is a functional differential equation, for which explicit solutions and convergence criteria are not known (for most implementations of $g(s)$). In the gamma model, the signals $\int_0^t ds\, g_k(s) y(t-s)$ are computed by the system and locally available as $y_k(t)$ at the synaptic junctions $W_k$. Thus, in the gamma model, (12) reduces to -

$$\frac{dW_k}{dt} = \eta x(t) y_k^T(t) . \qquad (13)$$

This learning rule encodes simultaneous correlations (for k=0) as well as temporal associations (for $k \geq 1$). Since the gamma Hebb rule is structurally similar to the conventional Hebb rule, it is also local both in time and space.

## 4    RELATION TO OTHER MODELS

The gamma model is related to Tank and Hopfield 's CITN model in that both models decompose W(t) into a linear combination of gamma kernels. The weights in the CITN system are preset and fixed. The gamma model, expressed as a regular additive system, allows conventional adaptation procedures to train the system parameters; $\mu$ and K adapt the depth and shape of the memory, while $W_0,..,W_K$ encode spatiotemporal correlations between neural states.

Time-Delay-Neural-Nets (TDNN) are characterized by a tapped delay line memory structure. The relation is best illustrated by an example. Consider a linear one-layer

feedforward convolution model, described by -

$$x(t) = e(t)$$

$$y(t) = \int_0^t W(t-s)\, x(s)\, ds \qquad (14)$$

where $x(t)$, $e(t)$ and $y(t)$ are N-dimensional signals and $W(t)$ a $N \times N \times [0, \infty]$ dimensional weight matrix. This system can be approximated in discrete time by -

$$x(n) = e(n)$$

$$y(n) = \sum_{m=0}^{n} W(n-m)\, x(m) \qquad (15)$$

which is the TDNN formulation. An alternative approximation of the convolution model by means of a (discrete-time) gamma model, is described by (figure 1) -

$$x_0(n) = e(n)$$

$$x_k(n) = (1-\mu)\, x_k(n-1) + \mu x_{k-1}(n-1) \quad k=1,..,K$$

$$y(n) = \sum_{k=0}^{K} W_k x_k(n) \qquad (16)$$

The recursive memory structure in the gamma model is stable for $0 \leq \mu \leq 2$, but an interesting memory structure is obtained only for $0 < \mu \leq 1$. For $\mu = 0$, this system collapses to a static additive net. In this case, no information from past signal values are stored in the net. For $0 < \mu < 1$, the system works as a discrete-time CITN. The gamma memory structure consists of a cascade of first-order leaky integrators. Since the total memory structure is of order K, the shape of the memory is not restricted to a recency gradient. The effective memory depth approximates $\dfrac{K}{\mu}$ for small $\mu$. For $\mu = 1$, the gamma model becomes a TDNN. In this case, memory is implemented by a tapped delay line. The strength of the gamma model is that the parameters $\mu$ and $K$ can be adapted by conventional additive learning procedures. Thus, the optimal temporal structure of the neural system, whether of CITN or TDNN type, is part of the training phase in a gamma neural net. Finally, the application of the gamma memory structure is of course not limited to one-layer feedforward systems. The topologies suggested by Jordan (1986) and Elman (1988) can easily be extended to include gamma memory.

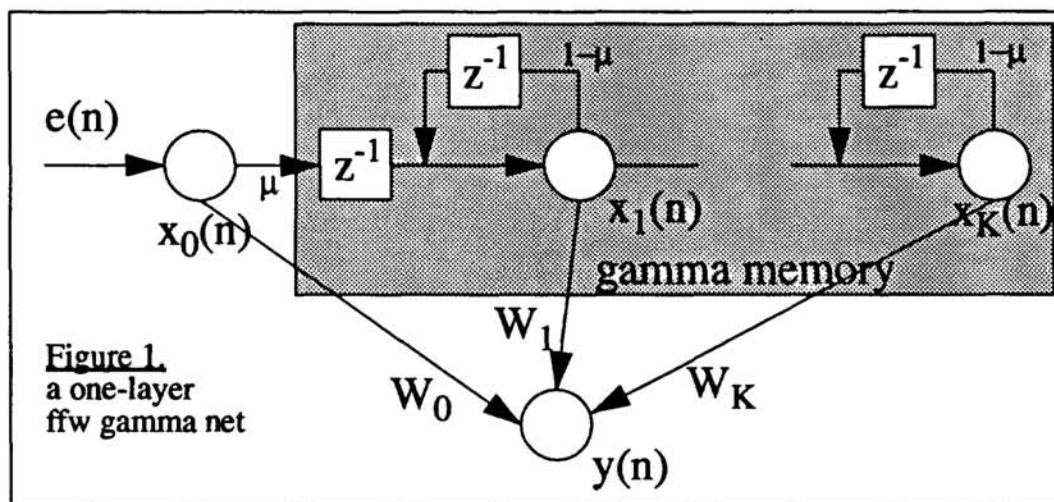

**Figure 1.**
a one-layer
ffw gamma net

## 5    CONCLUSIONS

We have introduced the gamma neural model, a neural net model for temporal processing, that generalizes most existing approaches, such as the CITN and TDNN models. The model can be described as a conventional dynamic additive model, enabling direct application of existing learning procedures for additive models. In the gamma model, dynamic objects are encoded by the same learning equations as static objects.

### Acknowledgments

This work has been partially supported by NSF grants ECS-8915218 and DDM-8914084.

### References

Cohen et. al., 1979. Stable oscillations in single species growth models with hereditary effects. *Mathematical Biosciences* 44:255-268, 1979.

DeVries and Principe, 1990. The gamma neural net - A new model for temporal processing. submitted to *Neural Networks,* Nov.1990.

Elman, 1988. Finding structure in time. *CRL technical report 8801*, 1988.

Jordan, 1986. Attractor dynamics and parallelism in a connectionist sequential machine. *Proc. Cognitive Science* 1986.

Lang et. al. 1990. A time-delay neural network architecture for isolated word recognition. *Neural Networks, vol.3 (1)*, 1990.

Tank and Hopfield, 1987. Concentrating information in time: analog neural networks with applications to speech recognition problems. *1st int. conf. on neural networks,* IEEE, 1987.